# Maximum Likelihood Estimation of Intrinsic Dimension

**Elizaveta Levina**
Department of Statistics
University of Michigan
Ann Arbor MI 48109-1092
elevina@umich.edu

**Peter J. Bickel**
Department of Statistics
University of California
Berkeley CA 94720-3860
bickel@stat.berkeley.edu

## Abstract

We propose a new method for estimating intrinsic dimension of a dataset derived by applying the principle of maximum likelihood to the distances between close neighbors. We derive the estimator by a Poisson process approximation, assess its bias and variance theoretically and by simulations, and apply it to a number of simulated and real datasets. We also show it has the best overall performance compared with two other intrinsic dimension estimators.

## 1 Introduction

There is a consensus in the high-dimensional data analysis community that the only reason any methods work in very high dimensions is that, in fact, the data are not truly high-dimensional. Rather, they are embedded in a high-dimensional space, but can be efficiently summarized in a space of a much lower dimension, such as a nonlinear manifold. Then one can reduce dimension without losing much information for many types of real-life high-dimensional data, such as images, and avoid many of the "curses of dimensionality". Learning these data manifolds can improve performance in classification and other applications, but if the data structure is complex and nonlinear, dimensionality reduction can be a hard problem.

Traditional methods for dimensionality reduction include principal component analysis (PCA), which only deals with linear projections of the data, and multidimensional scaling (MDS), which aims at preserving pairwise distances and traditionally is used for visualizing data. Recently, there has been a surge of interest in manifold projection methods (Locally Linear Embedding (LLE) [1], Isomap [2], Laplacian and Hessian Eigenmaps [3, 4], and others), which focus on finding a nonlinear low-dimensional embedding of high-dimensional data. So far, these methods have mostly been used for exploratory tasks such as visualization, but they have also been successfully applied to classification problems [5, 6].

The dimension of the embedding is a key parameter for manifold projection methods: if the dimension is too small, important data features are "collapsed" onto the same dimension, and if the dimension is too large, the projections become noisy and, in some cases, unstable. There is no consensus, however, on how this dimension should be determined. LLE [1] and its variants assume the manifold dimension

is provided by the user. Isomap [2] provides error curves that can be "eyeballed" to estimate dimension. The charting algorithm, a recent LLE variant [7], uses a heuristic estimate of dimension which is essentially equivalent to the regression estimator of [8] discussed below. Constructing a reliable estimator of intrinsic dimension and understanding its statistical properties will clearly facilitate further applications of manifold projection methods and improve their performance.

We note that for applications such as classification, cross-validation is in principle the simplest solution – just pick the dimension which gives the lowest classification error. However, in practice the computational cost of cross-validating for the dimension is prohibitive, and an estimate of the intrinsic dimension will still be helpful, either to be used directly or to narrow down the range for cross-validation.

In this paper, we present a new estimator of intrinsic dimension, study its statistical properties, and compare it to other estimators on both simulated and real datasets. Section 2 reviews previous work on intrinsic dimension. In Section 3 we derive the estimator and give its approximate asymptotic bias and variance. Section 4 presents results on datasets and compares our estimator to two other estimators of intrinsic dimension. Section 5 concludes with discussion.

## 2   Previous Work on Intrinsic Dimension Estimation

The existing approaches to estimating the intrinsic dimension can be roughly divided into two groups: eigenvalue or projection methods, and geometric methods. Eigenvalue methods, from the early proposal of [9] to a recent variant [10] are based on a global or local PCA, with intrinsic dimension determined by the number of eigenvalues greater than a given threshold. Global PCA methods fail on nonlinear manifolds, and local methods depend heavily on the precise choice of local regions and thresholds [11]. The eigenvalue methods may be a good tool for exploratory data analysis, where one might plot the eigenvalues and look for a clear-cut boundary, but not for providing reliable estimates of intrinsic dimension.

The geometric methods exploit the intrinsic geometry of the dataset and are most often based on fractal dimensions or nearest neighbor (NN) distances. Perhaps the most popular fractal dimension is the correlation dimension [12, 13]: given a set $S_n = \{x_1, \ldots, x_n\}$ in a metric space, define

$$C_n(r) = \frac{2}{n(n-1)} \sum_{i=1}^{n} \sum_{j=i+1}^{n} \mathbf{1}\{\|x_i - x_j\| < r\}. \tag{1}$$

The correlation dimension is then estimated by plotting $\log C_n(r)$ against $\log r$ and estimating the slope of the linear part [12]. A recent variant [13] proposed plotting this estimate against the true dimension for some simulated data and then using this calibrating curve to estimate the dimension of a new dataset. This requires a different curve for each $n$, and the choice of calibration data may affect performance. The capacity dimension and packing numbers have also been used [14]. While the fractal methods successfully exploit certain geometric aspects of the data, the statistical properties of these methods have not been studied.

The correlation dimension (1) implicitly uses NN distances, and there are methods that focus on them explicitly. The use of NN distances relies on the following fact: if $X_1, \ldots, X_n$ are an independent identically distributed (i.i.d.) sample from a density $f(x)$ in $\mathbb{R}^m$, and $T_k(x)$ is the Euclidean distance from a fixed point $x$ to its $k$-th NN in the sample, then

$$\frac{k}{n} \approx f(x) V(m) [T_k(x)]^m, \tag{2}$$

where $V(m) = \pi^{m/2}[\Gamma(m/2+1)]^{-1}$ is the volume of the unit sphere in $\mathbb{R}^m$. That is, the proportion of sample points falling into a ball around $x$ is roughly $f(x)$ times the volume of the ball.

The relationship (2) can be used to estimate the dimension by regressing $\log \bar{T}_k$ on $\log k$ over a suitable range of $k$, where $\bar{T}_k = n^{-1} \sum_{i=1}^n T_k(X_i)$ is the average of distances from each point to its $k$-th NN [8, 11]. A comparison of this method to a local eigenvalue method [11] found that the NN method suffered more from underestimating dimension for high-dimensional datasets, but the eigenvalue method was sensitive to noise and parameter settings. A more sophisticated NN approach was recently proposed in [15], where the dimension is estimated from the length of the minimal spanning tree on the geodesic NN distances computed by Isomap.

While there are certainly existing methods available for estimating intrinsic dimension, there are some issues that have not been adequately addressed. The behavior of the estimators as a function of sample size and dimension is not well understood or studied beyond the obvious "curse of dimensionality"; the statistical properties of the estimators, such as bias and variance, have not been looked at (with the exception of [15]); and comparisons between methods are not always presented.

## 3    A Maximum Likelihood Estimator of Intrinsic Dimension

Here we derive the maximum likelihood estimator (MLE) of the dimension $m$ from i.i.d. observations $X_1, \ldots, X_n$ in $\mathbb{R}^p$. The observations represent an embedding of a lower-dimensional sample, i.e., $X_i = g(Y_i)$, where $Y_i$ are sampled from an unknown smooth density $f$ on $\mathbb{R}^m$, with unknown $m \leq p$, and $g$ is a continuous and sufficiently smooth (but not necessarily globally isometric) mapping. This assumption ensures that close neighbors in $\mathbb{R}^m$ are mapped to close neighbors in the embedding.

The basic idea is to fix a point $x$, assume $f(x) \approx$ const in a small sphere $S_x(R)$ of radius $R$ around $x$, and treat the observations as a homogeneous Poisson process in $S_x(R)$. Consider the inhomogeneous process $\{N(t,x), 0 \leq t \leq R\}$,

$$N(t,x) = \sum_{i=1}^n \mathbf{1}\{X_i \in S_x(t)\} \tag{3}$$

which counts observations within distance $t$ from $x$. Approximating this binomial (fixed $n$) process by a Poisson process and suppressing the dependence on $x$ for now, we can write the rate $\lambda(t)$ of the process $N(t)$ as

$$\lambda(t) = f(x)V(m)mt^{m-1} \tag{4}$$

This follows immediately from the Poisson process properties since $V(m)mt^{m-1} = \frac{d}{dt}[V(m)t^m]$ is the surface area of the sphere $S_x(t)$. Letting $\theta = \log f(x)$, we can write the log-likelihood of the observed process $N(t)$ as (see e.g., [16])

$$L(m,\theta) = \int_0^R \log \lambda(t)\, dN(t) - \int_0^R \lambda(t)\, dt$$

This is an exponential family for which MLEs exist with probability $\to 1$ as $n \to \infty$ and are unique. The MLEs must satisfy the likelihood equations

$$\frac{\partial L}{\partial \theta} = \int_0^R dN(t) - \int_0^R \lambda(t)dt = N(R) - e^\theta V(m)R^m = 0, \tag{5}$$

$$\begin{aligned}\frac{\partial L}{\partial m} = &\left(\frac{1}{m} + \frac{V'(m)}{V(m)}\right)N(R) + \int_0^R \log t\, dN(t) - \\ &- e^\theta V(m)R^m \left(\log R + \frac{V'(m)}{V(m)}\right) = 0.\end{aligned} \tag{6}$$

Substituting (5) into (6) gives the MLE for $m$:

$$\hat{m}_R(x) = \left[ \frac{1}{N(R,x)} \sum_{j=1}^{N(R,x)} \log \frac{R}{T_j(x)} \right]^{-1} . \tag{7}$$

In practice, it may be more convenient to fix the number of neighbors $k$ rather than the radius of the sphere $R$. Then the estimate in (7) becomes

$$\hat{m}_k(x) = \left[ \frac{1}{k-1} \sum_{j=1}^{k-1} \log \frac{T_k(x)}{T_j(x)} \right]^{-1} . \tag{8}$$

Note that we omit the last (zero) term in the sum in (7). One could divide by $k-2$ rather than $k-1$ to make the estimator asymptotically unbiased, as we show below. Also note that the MLE of $\theta$ can be used to obtain an instant estimate of the entropy of $f$, which was also provided by the method used in [15].

For some applications, one may want to evaluate local dimension estimates at every data point, or average estimated dimensions within data clusters. We will, however, assume that all the data points come from the same "manifold", and therefore average over all observations.

The choice of $k$ clearly affects the estimate. It can be the case that a dataset has different intrinsic dimensions at different scales, e.g., a line with noise added to it can be viewed as either 1-d or 2-d (this is discussed in detail in [14]). In such a case, it is informative to have different estimates at different scales. In general, for our estimator to work well the sphere should be small and contain sufficiently many points, and we have work in progress on choosing such a $k$ automatically. For this paper, though, we simply average over a range of small to moderate values $k = k_1 \ldots k_2$ to get the final estimates

$$\hat{m}_k = \frac{1}{n} \sum_{i=1}^{n} \hat{m}_k(X_i) , \qquad \hat{m} = \frac{1}{k_2 - k_1 + 1} \sum_{k=k_1}^{k_2} \hat{m}_k . \tag{9}$$

The choice of $k_1$ and $k_2$ and behavior of $\hat{m}_k$ as a function of $k$ are discussed further in Section 4. The only parameters to set for this method are $k_1$ and $k_2$, and the computational cost is essentially the cost of finding $k_2$ nearest neighbors for every point, which has to be done for most manifold projection methods anyway.

## 3.1 Asymptotic behavior of the estimator for $m$ fixed, $n \to \infty$.

Here we give a sketchy discussion of the asymptotic bias and variance of our estimator, to be elaborated elsewhere. The computations here are under the assumption that $m$ is fixed, $n \to \infty, k \to \infty$, and $k/n \to 0$.

As we remarked, for a given $x$ if $n \to \infty$ and $R \to 0$, the inhomogeneous binomial process $N(t,x)$ in (3) converges weakly to the inhomogeneous Poisson process with rate $\lambda(t)$ given by (4). If we condition on the distance $T_k(x)$ and assume the Poisson approximation is exact, then $\{m^{-1}\log(T_k/T_j) : 1 \le j \le k-1\}$ are distributed as the order statistics of a sample of size $k-1$ from a standard exponential distribution. Hence $U = m^{-1} \sum_{j=1}^{k-1} \log(T_k/T_j)$ has a $\mathrm{Gamma}(k-1,1)$ distribution, and $EU^{-1} = 1/(k-2)$. If we use $k-2$ to normalize, then under these assumptions, to a first order approximation

$$E\left(\hat{m}_k(x)\right) = m, \quad \mathrm{Var}\left(\hat{m}_k(x)\right) = \frac{m^2}{k-3} \tag{10}$$

As this analysis is asymptotic in both $k$ and $n$, the factor $(k-1)/(k-2)$ makes no difference. There are, of course, higher order terms since $N(t,x)$ is in fact a binomial process with $EN(t,x) = \lambda(t)\left(1 + O(t^2)\right)$, where $O(t^2)$ depends on $m$.

With approximations (10), we have $E\hat{m} = E\hat{m}_k = m$, but the computation of $\text{Var}(\hat{m})$ is complicated by the dependence among $\hat{m}_k(X_i)$. We have a heuristic argument (omitted for lack of space) that, by dividing $\hat{m}_k(X_i)$ into $n/k$ roughly independent groups of size $k$ each, the variance can be shown to be of order $n^{-1}$, as it would if the estimators were independent. Our simulations confirm that this approximation is reasonable – for instance, for $m$-d Gaussians the ratio of the theoretical $\text{SD} = C(k_1, k_2)m/\sqrt{n}$ (where $C(k_1, k_2)$ is calculated as if all the terms in (9) were independent) to the actual SD of $\hat{m}$ was between 0.7 and 1.3 for the range of values of $m$ and $n$ considered in Section 4. The bias, however, behaves worse than the asymptotics predict, as we discuss further in Section 5.

## 4  Numerical Results

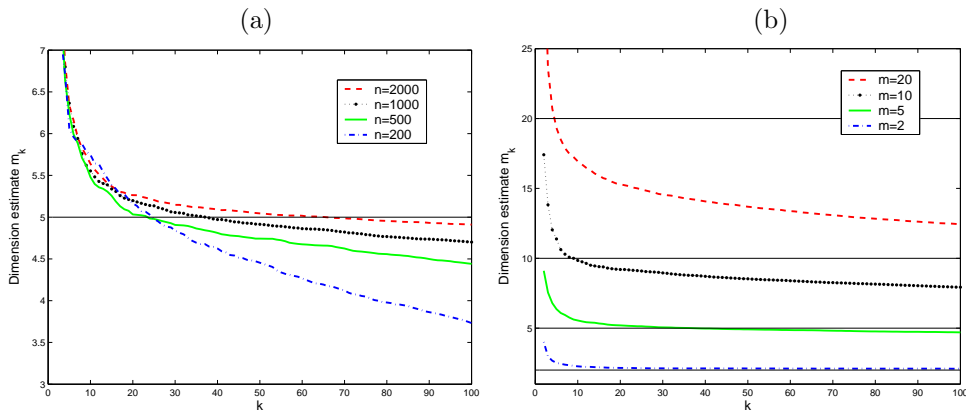

Figure 1: The estimator $\hat{m}_k$ as a function of $k$. (a) 5-dimensional normal for several sample sizes. (b) Various $m$-dimensional normals with sample size $n = 1000$.

We first investigate the properties of our estimator in detail by simulations, and then apply it to real datasets. The first issue is the behavior of $\hat{m}_k$ as a function of $k$. The results shown in Fig. 1 are for $m$-d Gaussians $N_m(0, I)$, and a similar pattern holds for observations in a unit cube, on a hypersphere, and on the popular "Swiss roll" manifold. Fig. 1(a) shows $\hat{m}_k$ for a 5-d Gaussian as a function of $k$ for several sample sizes $n$. For very small $k$ the approximation does not work yet and $\hat{m}_k$ is unreasonably high, but for $k$ as small as 10, the estimate is near the true value $m = 5$. The estimate shows some negative bias for large $k$, which decreases with growing sample size $n$, and, as Fig. 1(b) shows, increases with dimension. Note, however, that it is the intrinsic dimension $m$ rather than the embedding dimension $p \geq m$ that matters; and as our examples below and many examples elsewhere show, the intrinsic dimension for real data is frequently low.

The plots in Fig. 1 show that the "ideal" range $k_1 \ldots k_2$ is different for every combination of $m$ and $n$, but the estimator is fairly stable as a function of $k$, apart from the first few values. While fine-tuning the range $k_1 \ldots k_2$ for different $n$ is possible and would reduce the bias, for simplicity and reproducibility of our results we fix $k_1 = 10$, $k_2 = 20$ throughout this paper. In this range, the estimates are not

affected much by sample size or the positive bias for very small $k$, at least for the range of $m$ and $n$ under consideration.

Next, we investigate an important and often overlooked issue of what happens when the data are near a manifold as opposed to exactly on a manifold. Fig. 2(a) shows simulation results for a 5-d correlated Gaussian with mean 0, and covariance matrix $[\sigma_{ij}] = [\rho + (1 - \rho)\delta_{ij}]$, with $\delta_{ij} = \mathbf{1}\{i = j\}$. As $\rho$ changes from 0 to 1, the dimension changes from 5 (full spherical Gaussian) to 1 (a line in $\mathbb{R}^5$), with intermediate values of $\rho$ providing noisy versions.

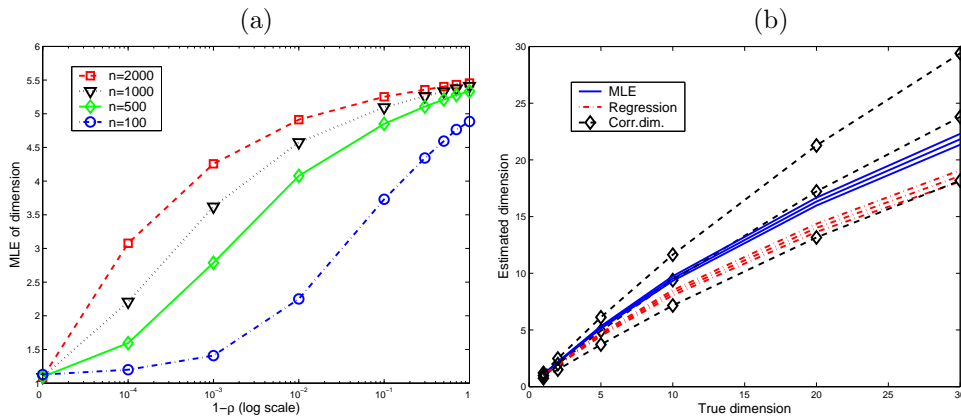

Figure 2: (a) Data near a manifold: estimated dimension for correlated 5-d normal as a function of $1 - \rho$. (b) The MLE, regression, and correlation dimension for uniform distributions on spheres with $n = 1000$. The three lines for each method show the mean $\pm 2$ SD (95% confidence intervals) over 1000 replications.

The plots in Fig. 2(a) show that the MLE of dimension does not drop unless $\rho$ is very close to 1, so the estimate is not affected by whether the data cloud is spherical or elongated. For $\rho$ close to 1, when the dimension really drops, the estimate depends significantly on the sample size, which is to be expected: $n = 100$ highly correlated points look like a line, but $n = 2000$ points fill out the space around the line. This highlights the fundamental dependence of intrinsic dimension on the neighborhood scale, particularly when the data may be observed with noise. The MLE of dimension, while reflecting this dependence, behaves reasonably and robustly as a function of both $\rho$ and $n$.

A comparison of the MLE, the regression estimator (regressing $\log \overline{T}_k$ on $\log k$), and the correlation dimension is shown in Fig. 2(b). The comparison is shown on uniformly distributed points on the surface of an $m-$dimensional sphere, but a similar pattern held in all our simulations. The regression range was held at $k = 10 \ldots 20$ (the same as the MLE) for fair comparison, and the regression for correlation dimension was based on the first $10 \ldots 100$ distinct values of $\log C_n(r)$, to reflect the fact there are many more points for the $\log C_n(r)$ regression than for the $\log \overline{T}_k$ regression. We found in general that the correlation dimension graph can have more than one linear part, and is more sensitive to the choice of range than either the MLE or the regression estimator, but we tried to set the parameters for all methods in a way that does not give an unfair advantage to any and is easily reproducible.

The comparison shows that, while all methods suffer from negative bias for higher dimensions, the correlation dimension has the smallest bias, with the MLE coming

in close second. However, the variance of correlation dimension is much higher than that of the MLE (the SD is at least 10 times higher for *all* dimensions). The regression estimator, on the other hand, has relatively low variance (though always higher than the MLE) but the largest negative bias. On the balance of bias and variance, MLE is clearly the best choice.

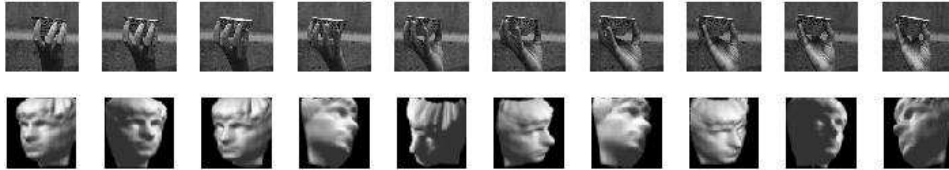

Figure 3: Two image datasets: hand rotation and Isomap faces (example images).

Table 1: Estimated dimensions for popular manifold datasets. For the Swiss roll, the table gives mean(SD) over 1000 uniform samples.

| Dataset | Data dim. | Sample size | MLE | Regression | Corr. dim. |
|---|---|---|---|---|---|
| Swiss roll | 3 | 1000 | 2.1(0.02) | 1.8(0.03) | 2.0(0.24) |
| Faces | $64 \times 64$ | 698 | 4.3 | 4.0 | 3.5 |
| Hands | $480 \times 512$ | 481 | 3.1 | 2.5 | $3.9^1$ |

Finally, we compare the estimators on three popular manifold datasets (Table 1): the Swiss roll, and two image datasets shown on Fig. 3: the Isomap face database[2], and the hand rotation sequence[3] used in [14]. For the Swiss roll, the MLE again provides the best combination of bias and variance.

The face database consists of images of an artificial face under three changing conditions: illumination, and vertical and horizontal orientation. Hence the intrinsic dimension of the dataset should be 3, but only if we had the full 3-d images of the face. All we have, however, are 2-d projections of the face, and it is clear that one needs more than one "basis" image to represent different poses (from casual inspection, front view and profile seem sufficient). The estimated dimension of about 4 is therefore very reasonable.

The hand image data is a real video sequence of a hand rotating along a 1-d curve in space, but again several basis 2-d images are needed to represent different poses (in this case, front, back, and profile seem sufficient). The estimated dimension around 3 therefore seems reasonable. We note that the correlation dimension provides two completely different answers for this dataset, depending on which linear part of the curve is used; this is further evidence of its high variance, which makes it a less reliable estimate that the MLE.

## 5 Discussion

In this paper, we have derived a maximum likelihood estimator of intrinsic dimension and some asymptotic approximations to its bias and variance. We have shown

that the MLE produces good results on a range of simulated and real datasets and outperforms two other dimension estimators. It does, however, suffer from a negative bias for high dimensions, which is a problem shared by all dimension estimators. One reason for this is that our approximation is based on sufficiently many observations falling into a small sphere, and that requires very large sample sizes in high dimensions (we shall elaborate and quantify this further elsewhere). For some datasets, such as points in a unit cube, there is also the issue of edge effects, which generally become more severe in high dimensions. One can potentially reduce the negative bias by removing the edge points by some criterion, but we found that the edge effects are small compared to the sample size problem, and we have been unable to achieve significant improvement in this manner. Another option used by [13] is calibration on simulated datasets with known dimension, but since the bias depends on the sampling distribution, and a different curve would be needed for every sample size, calibration does not solve the problem either. One should keep in mind, however, that for most interesting applications intrinsic dimension will not be very high – otherwise there is not much benefit in dimensionality reduction; hence in practice the MLE will provide a good estimate of dimension most of the time.

## Footnotes

[1]This estimate is obtained from the range 500...1000. For this dataset, the correlation dimension curve has two distinct linear parts, with the first part over the range we would normally use, 10...100, producing dimension 19.7, which is clearly unreasonable.

[2]http://isomap.stanford.edu/datasets.html

[3]http://vasc.ri.cmu.edu//idb/html/motion/hand/index.html

## References

[1] S. T. Roweis and L. K. Saul. Nonlinear dimensionality reduction by locally linear embedding. *Science*, 290:2323–2326, 2000.

[2] J. B. Tenenbaum, V. de Silva, and J. C. Landford. A global geometric framework for nonlinear dimensionality reduction. *Science*, 290:2319–2323, 2000.

[3] M. Belkin and P. Niyogi. Laplacian eigenmaps and spectral techniques for embedding and clustering. In *Advances in NIPS*, volume 14. MIT Press, 2002.

[4] D. L. Donoho and C. Grimes. Hessian eigenmaps: New locally linear embedding techniques for high-dimensional data. Technical Report TR 2003-08, Department of Statistics, Stanford University, 2003.

[5] M. Belkin and P. Niyogi. Using manifold structure for partially labelled classification. In *Advances in NIPS*, volume 15. MIT Press, 2003.

[6] M. Vlachos, C. Domeniconi, D. Gunopulos, G. Kollios, and N. Koudas. Non-linear dimensionality reduction techniques for classification and visualization. In *Proceedings of 8th SIGKDD*, pages 645–651. Edmonton, Canada, 2002.

[7] M. Brand. Charting a manifold. In *Advances in NIPS*, volume 14. MIT Press, 2002.

[8] K.W. Pettis, T.A. Bailey, A.K. Jain, and R.C. Dubes. An intrinsic dimensionality estimator from near-neighbor information. *IEEE Trans. on PAMI*, 1:25–37, 1979.

[9] K. Fukunaga and D.R. Olsen. An algorithm for finding intrinsic dimensionality of data. *IEEE Trans. on Computers*, C-20:176–183, 1971.

[10] J. Bruske and G. Sommer. Intrinsic dimensionality estimation with optimally topology preserving maps. *IEEE Trans. on PAMI*, 20(5):572–575, 1998.

[11] P. Verveer and R. Duin. An evaluation of intrinsic dimensionality estimators. *IEEE Trans. on PAMI*, 17(1):81–86, 1995.

[12] P. Grassberger and I. Procaccia. Measuring the strangeness of strange attractors. *Physica*, D9:189–208, 1983.

[13] F. Camastra and A. Vinciarelli. Estimating the intrinsic dimension of data with a fractal-based approach. *IEEE Trans. on PAMI*, 24(10):1404–1407, 2002.

[14] B. Kegl. Intrinsic dimension estimation using packing numbers. In *Advances in NIPS*, volume 14. MIT Press, 2002.

[15] J. Costa and A. O. Hero. Geodesic entropic graphs for dimension and entropy estimation in manifold learning. *IEEE Trans. on Signal Processing*, 2004. To appear.

[16] D. L. Snyder. *Random Point Processes*. Wiley, New York, 1975.
